# Divide-and-Conquer Matrix Factorization

**Lester Mackey**[a]     **Ameet Talwalkar**[a]     **Michael I. Jordan**[a, b]

[a] Department of Electrical Engineering and Computer Science, UC Berkeley
[b] Department of Statistics, UC Berkeley

## Abstract

This work introduces Divide-Factor-Combine (DFC), a parallel divide-and-conquer framework for noisy matrix factorization. DFC divides a large-scale matrix factorization task into smaller subproblems, solves each subproblem in parallel using an arbitrary base matrix factorization algorithm, and combines the subproblem solutions using techniques from randomized matrix approximation. Our experiments with collaborative filtering, video background modeling, and simulated data demonstrate the near-linear to super-linear speed-ups attainable with this approach. Moreover, our analysis shows that DFC enjoys high-probability recovery guarantees comparable to those of its base algorithm.

## 1 Introduction

The goal in matrix factorization is to recover a low-rank matrix from irrelevant noise and corruption. We focus on two instances of the problem: noisy matrix completion, i.e., recovering a low-rank matrix from a small subset of noisy entries, and noisy robust matrix factorization [2, 3, 4], i.e., recovering a low-rank matrix from corruption by noise and outliers of arbitrary magnitude. Examples of the matrix completion problem include collaborative filtering for recommender systems, link prediction for social networks, and click prediction for web search, while applications of robust matrix factorization arise in video surveillance [2], graphical model selection [4], document modeling [17], and image alignment [21].

These two classes of matrix factorization problems have attracted significant interest in the research community. In particular, convex formulations of noisy matrix factorization have been shown to admit strong theoretical recovery guarantees [1, 2, 3, 20], and a variety of algorithms (e.g., [15, 16, 23]) have been developed for solving both matrix completion and robust matrix factorization via convex relaxation. Unfortunately, these methods are inherently sequential and all rely on the repeated and costly computation of truncated SVDs, factors that limit the scalability of the algorithms.

To improve scalability and leverage the growing availability of parallel computing architectures, we propose a divide-and-conquer framework for large-scale matrix factorization. Our framework, entitled Divide-Factor-Combine (DFC), randomly divides the original matrix factorization task into cheaper subproblems, solves those subproblems in parallel using any base matrix factorization algorithm, and combines the solutions to the subproblem using efficient techniques from randomized matrix approximation. The inherent parallelism of DFC allows for near-linear to superlinear speed-ups in practice, while our theory provides high-probability recovery guarantees for DFC comparable to those enjoyed by its base algorithm.

The remainder of the paper is organized as follows. In Section 2, we define the setting of noisy matrix factorization and introduce the components of the DFC framework. To illustrate the significant speed-up and robustness of DFC and to highlight the effectiveness of DFC ensembling, we present experimental results on collaborative filtering, video background modeling, and simulated data in Section 3. Our theoretical analysis follows in Section 4. There, we establish high-probability noisy recovery guarantees for DFC that rest upon a novel analysis of randomized matrix approximation and a new recovery result for noisy matrix completion.

**Notation** For $\mathbf{M} \in \mathbb{R}^{m \times n}$, we define $\mathbf{M}_{(i)}$ as the $i$th row vector and $\mathbf{M}_{ij}$ as the $ij$th entry. If $\mathrm{rank}(\mathbf{M}) = r$, we write the compact singular value decomposition (SVD) of $\mathbf{M}$ as $\mathbf{U}_M \boldsymbol{\Sigma}_M \mathbf{V}_M^\top$, where $\boldsymbol{\Sigma}_M$ is diagonal and contains the $r$ non-zero singular values of $\mathbf{M}$, and $\mathbf{U}_M \in \mathbb{R}^{m \times r}$ and $\mathbf{V}_M \in \mathbb{R}^{n \times r}$ are the corresponding left and right singular vectors of $\mathbf{M}$. We define $\mathbf{M}^+ = \mathbf{V}_M \boldsymbol{\Sigma}_M^{-1} \mathbf{U}_M^\top$ as the Moore-Penrose pseudoinverse of $\mathbf{M}$ and $\mathbf{P}_M = \mathbf{M}\mathbf{M}^+$ as the orthogonal projection onto the column space of $\mathbf{M}$. We let $\|\cdot\|_2$, $\|\cdot\|_F$, and $\|\cdot\|_*$ respectively denote the spectral, Frobenius, and nuclear norms of a matrix and let $\|\cdot\|$ represent the $\ell_2$ norm of a vector.

## 2 The Divide-Factor-Combine Framework

In this section, we present our divide-and-conquer framework for scalable noisy matrix factorization. We begin by defining the problem setting of interest.

### 2.1 Noisy Matrix Factorization (MF)

In the setting of noisy matrix factorization, we observe a subset of the entries of a matrix $\mathbf{M} = \mathbf{L}_0 + \mathbf{S}_0 + \mathbf{Z}_0 \in \mathbb{R}^{m \times n}$, where $\mathbf{L}_0$ has rank $r \ll m, n$, $\mathbf{S}_0$ represents a sparse matrix of outliers of arbitrary magnitude, and $\mathbf{Z}_0$ is a dense noise matrix. We let $\Omega$ represent the locations of the observed entries and $\mathcal{P}_\Omega$ be the orthogonal projection onto the space of $m \times n$ matrices with support $\Omega$, so that

$$(\mathcal{P}_\Omega(\mathbf{M}))_{ij} = \mathbf{M}_{ij}, \text{ if } (i,j) \in \Omega \quad \text{and} \quad (\mathcal{P}_\Omega(\mathbf{M}))_{ij} = 0 \text{ otherwise.}$$

Our goal is to recover the low-rank matrix $\mathbf{L}_0$ from $\mathcal{P}_\Omega(\mathbf{M})$ with error proportional to the noise level $\Delta \triangleq \|\mathbf{Z}_0\|_F$. We will focus on two specific instances of this general problem:

- **Noisy Matrix Completion (MC):** $s \triangleq |\Omega|$ entries of $\mathbf{M}$ are revealed uniformly without replacement, along with their locations. There are no outliers, so that $\mathbf{S}_0$ is identically zero.

- **Noisy Robust Matrix Factorization (RMF):** $\mathbf{S}_0$ is identically zero save for $s$ outlier entries of arbitrary magnitude with unknown locations distributed uniformly without replacement. All entries of $\mathbf{M}$ are observed, so that $\mathcal{P}_\Omega(\mathbf{M}) = \mathbf{M}$.

### 2.2 Divide-Factor-Combine

Algorithms 1 and 2 summarize two canonical examples of the general Divide-Factor-Combine framework that we refer to as DFC-PROJ and DFC-NYS. Each algorithm has three simple steps:

**(D step) Divide input matrix into submatrices:** DFC-PROJ randomly partitions $\mathcal{P}_\Omega(\mathbf{M})$ into $t$ $l$-column submatrices, $\{\mathcal{P}_\Omega(\mathbf{C}_1), \ldots, \mathcal{P}_\Omega(\mathbf{C}_t)\}$[1], while DFC-NYS selects an $l$-column submatrix, $\mathcal{P}_\Omega(\mathbf{C})$, and a $d$-row submatrix, $\mathcal{P}_\Omega(\mathbf{R})$, uniformly at random.

**(F step) Factor each submatrix in parallel using any base MF algorithm:** DFC-PROJ performs $t$ parallel submatrix factorizations, while DFC-NYS performs two such parallel factorizations. Standard base MF algorithms output the low-rank approximations $\{\hat{\mathbf{C}}_1, \ldots, \hat{\mathbf{C}}_t\}$ for DFC-PROJ and $\hat{\mathbf{C}}$, and $\hat{\mathbf{R}}$ for DFC-NYS. All matrices are retained in factored form.

**(C step) Combine submatrix estimates:** DFC-PROJ generates a final low-rank estimate $\hat{\mathbf{L}}^{proj}$ by projecting $[\hat{\mathbf{C}}_1, \ldots, \hat{\mathbf{C}}_t]$ onto the column space of $\hat{\mathbf{C}}_1$, while DFC-NYS forms the low-rank estimate $\hat{\mathbf{L}}^{nys}$ from $\hat{\mathbf{C}}$ and $\hat{\mathbf{R}}$ via the generalized Nyström method. These matrix approximation techniques are described in more detail in Section 2.3.

### 2.3 Randomized Matrix Approximations

Our divide-and-conquer algorithms rely on two methods that generate randomized low-rank approximations to an arbitrary matrix $\mathbf{M}$ from submatrices of $\mathbf{M}$.

| **Algorithm 1** DFC-PROJ | **Algorithm 2** DFC-NYS[a] |
|---|---|
| **Input:** $\mathcal{P}_\Omega(\mathbf{M}), t$ | **Input:** $\mathcal{P}_\Omega(\mathbf{M}), l, d$ |
| $\{\mathcal{P}_\Omega(\mathbf{C}_i)\}_{1 \le i \le t} = \text{SAMPCOL}(\mathcal{P}_\Omega(\mathbf{M}), t)$ | $\mathcal{P}_\Omega(\mathbf{C}), \mathcal{P}_\Omega(\mathbf{R}) = \text{SAMPCOLROW}(\mathcal{P}_\Omega(\mathbf{M}), l, d)$ |
| **do in parallel** | **do in parallel** |
| $\quad \hat{\mathbf{C}}_1 = \text{BASE-MF-ALG}(\mathcal{P}_\Omega(\mathbf{C}_1))$ | $\quad \hat{\mathbf{C}} = \text{BASE-MF-ALG}(\mathcal{P}_\Omega(\mathbf{C}))$ |
| $\qquad\qquad\vdots$ | $\quad \hat{\mathbf{R}} = \text{BASE-MF-ALG}(\mathcal{P}_\Omega(\mathbf{R}))$ |
| $\quad \hat{\mathbf{C}}_t = \text{BASE-MF-ALG}(\mathcal{P}_\Omega(\mathbf{C}_t))$ | **end do** |
| **end do** | $\hat{\mathbf{L}}^{nys} = \text{GENNYSTRÖM}(\hat{\mathbf{C}}, \hat{\mathbf{R}})$ |
| $\hat{\mathbf{L}}^{proj} = \text{COLPROJECTION}(\hat{\mathbf{C}}_1, \ldots, \hat{\mathbf{C}}_t)$ | |

[a]When $\mathbf{Q}$ is a submatrix of $\mathbf{M}$ we abuse notation and define $\mathcal{P}_\Omega(\mathbf{Q})$ as the corresponding submatrix of $\mathcal{P}_\Omega(\mathbf{M})$.

**Column Projection**   This approximation, introduced by Frieze et al. [7], is derived from column sampling of $\mathbf{M}$. We begin by sampling $l < n$ columns uniformly without replacement and let $\mathbf{C}$ be the $m \times l$ matrix of sampled columns. Then, column projection uses $\mathbf{C}$ to generate a "matrix projection" approximation [13] of $\mathbf{M}$ as follows:

$$\mathbf{L}^{proj} = \mathbf{C}\mathbf{C}^+\mathbf{M} = \mathbf{U}_C \mathbf{U}_C^\top \mathbf{M}.$$

In practice, we do not reconstruct $\mathbf{L}^{proj}$ but rather maintain low-rank factors, e.g., $\mathbf{U}_C$ and $\mathbf{U}_C^\top \mathbf{M}$.

**Generalized Nyström Method**   The standard Nyström method is often used to speed up large-scale learning applications involving symmetric positive semidefinite (SPSD) matrices [24] and has been generalized for arbitrary real-valued matrices [8]. In particular, after sampling columns to obtain $\mathbf{C}$, imagine that we independently sample $d < m$ rows uniformly without replacement. Let $\mathbf{R}$ be the $d \times n$ matrix of sampled rows and $\mathbf{W}$ be the $d \times l$ matrix formed from the intersection of the sampled rows and columns. Then, the generalized Nyström method uses $\mathbf{C}, \mathbf{W},$ and $\mathbf{R}$ to compute an "spectral reconstruction" approximation [13] of $\mathbf{M}$ as follows:

$$\mathbf{L}^{nys} = \mathbf{C}\mathbf{W}^+\mathbf{R} = \mathbf{C}\mathbf{V}_W \mathbf{\Sigma}_W^+ \mathbf{U}_W^\top \mathbf{R}.$$

As with $\mathbf{M}^{proj}$, we store low-rank factors of $\mathbf{L}^{nys}$, such as $\mathbf{C}\mathbf{V}_W \mathbf{\Sigma}_W^+$ and $\mathbf{U}_W^\top \mathbf{R}$.

## 2.4   Running Time of DFC

Many state-of-the-art MF algorithms have $\Omega(mnk_M)$ per-iteration time complexity due to the rank-$k_M$ truncated SVD performed on each iteration. DFC significantly reduces the per-iteration complexity to $\text{O}(mlk_{C_i})$ time for $\mathbf{C}_i$ (or $\mathbf{C}$) and $\text{O}(ndk_R)$ time for $\mathbf{R}$. The cost of combining the submatrix estimates is even smaller, since the outputs of standard MF algorithms are returned in factored form. Indeed, the column projection step of DFC-PROJ requires only $\text{O}(mk^2 + lk^2)$ time for $k \triangleq \max_i k_{C_i}$: $\text{O}(mk^2 + lk^2)$ time for the pseudoinversion of $\hat{\mathbf{C}}_1$ and $\text{O}(mk^2 + lk^2)$ time for matrix multiplication with each $\hat{\mathbf{C}}_i$ in parallel. Similarly, the generalized Nyström step of DFC-NYS requires only $\text{O}(l\bar{k}^2 + d\bar{k}^2 + \min(m,n)\bar{k}^2)$ time, where $\bar{k} \triangleq \max(k_C, k_R)$. Hence, DFC divides the expensive task of matrix factorization into smaller subproblems that can be executed in parallel and efficiently combines the low-rank, factored results.

## 2.5   Ensemble Methods

Ensemble methods have been shown to improve performance of matrix approximation algorithms, while straightforwardly leveraging the parallelism of modern many-core and distributed architectures [14]. As such, we propose ensemble variants of the DFC algorithms that demonstrably reduce recovery error while introducing a negligible cost to the parallel running time. For DFC-PROJ-ENS, rather than projecting only onto the column space of $\hat{\mathbf{C}}_1$, we project $[\hat{\mathbf{C}}_1, \ldots, \hat{\mathbf{C}}_t]$ onto the column space of each $\hat{\mathbf{C}}_i$ in parallel and then average the $t$ resulting low-rank approximations. For DFC-NYS-ENS, we choose a random $d$-row submatrix $\mathcal{P}_\Omega(\mathbf{R})$ as in DFC-NYS and independently partition the columns of $\mathcal{P}_\Omega(\mathbf{M})$ into $\{\mathcal{P}_\Omega(\mathbf{C}_1), \ldots, \mathcal{P}_\Omega(\mathbf{C}_t)\}$ as in DFC-PROJ. After running the

base MF algorithm on each submatrix, we apply the generalized Nyström method to each $(\hat{\mathbf{C}}_i, \hat{\mathbf{R}})$ pair in parallel and average the $t$ resulting low-rank approximations. Section 3 highlights the empirical effectiveness of ensembling.

## 3  Experimental Evaluation

We now explore the accuracy and speed-up of DFC on a variety of simulated and real-world datasets. We use state-of-the-art matrix factorization algorithms in our experiments: the Accelerated Proximal Gradient (APG) algorithm of [23] as our base noisy MC algorithm and the APG algorithm of [15] as our base noisy RMF algorithm. In all experiments, we use the default parameter settings suggested by [23] and [15], measure recovery error via root mean square error (RMSE), and report parallel running times for DFC. We moreover compare against two baseline methods: APG used on the full matrix $\mathbf{M}$ and PARTITION, which performs matrix factorization on $t$ submatrices just like DFC-PROJ but omits the final column projection step.

### 3.1  Simulations

For our simulations, we focused on square matrices ($m = n$) and generated random low-rank and sparse decompositions, similar to the schemes used in related work, e.g., [2, 12, 25]. We created $\mathbf{L}_0 \in \mathbb{R}^{m \times m}$ as a random product, $\mathbf{AB}^\top$, where $\mathbf{A}$ and $\mathbf{B}$ are $m \times r$ matrices with independent $\mathcal{N}(0, \sqrt{1/r})$ entries such that each entry of $\mathbf{L}_0$ has unit variance. $\mathbf{Z}_0$ contained independent $\mathcal{N}(0, 0.1)$ entries. In the MC setting, $s$ entries of $\mathbf{L}_0 + \mathbf{Z}_0$ were revealed uniformly at random. In the RMF setting, the support of $\mathbf{S}_0$ was generated uniformly at random, and the $s$ corrupted entries took values in $[0, 1]$ with uniform probability. For each algorithm, we report error between $\mathbf{L}_0$ and the recovered low-rank matrix, and all reported results are averages over five trials.

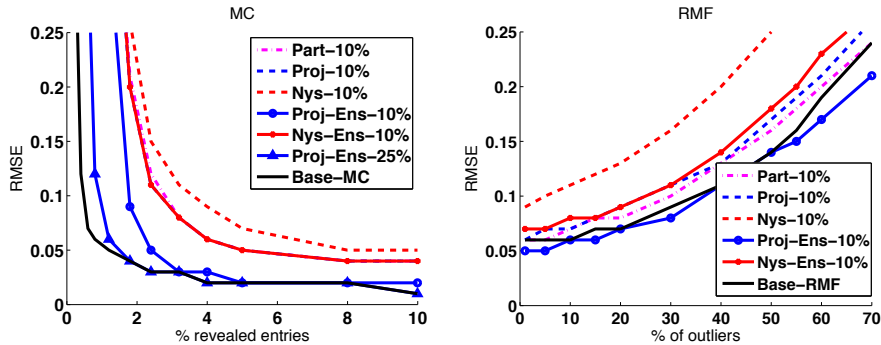

Figure 1: Recovery error of DFC relative to base algorithms.

We first explored the recovery error of DFC as a function of $s$, using ($m = 10K$, $r = 10$) with varying observation sparsity for MC and ($m = 1K$, $r = 10$) with a varying percentage of outliers for RMF. The results are summarized in Figure 1.[2] In both MC and RMF, the gaps in recovery between APG and DFC are small when sampling only 10% of rows and columns. Moreover, DFC-PROJ-ENS in particular consistently outperforms PARTITION and DFC-NYS-ENS and matches the performance of APG for most settings of $s$.

We next explored the speed-up of DFC as a function of matrix size. For MC, we revealed 4% of the matrix entries and set $r = 0.001 \cdot m$, while for RMF we fixed the percentage of outliers to 10% and set $r = 0.01 \cdot m$. We sampled 10% of rows and columns and observed that recovery errors were comparable to the errors presented in Figure 1 for similar settings of $s$; in particular, at all values of $n$ for both MC and RMF, the errors of APG and DFC-PROJ-ENS were nearly identical. Our timing results, presented in Figure 2, illustrate a near-linear speed-up for MC and a superlinear speed-up for RMF across varying matrix sizes. Note that the timing curves of the DFC algorithms and PARTITION all overlap, a fact that highlights the minimal computational cost of the final matrix approximation step.

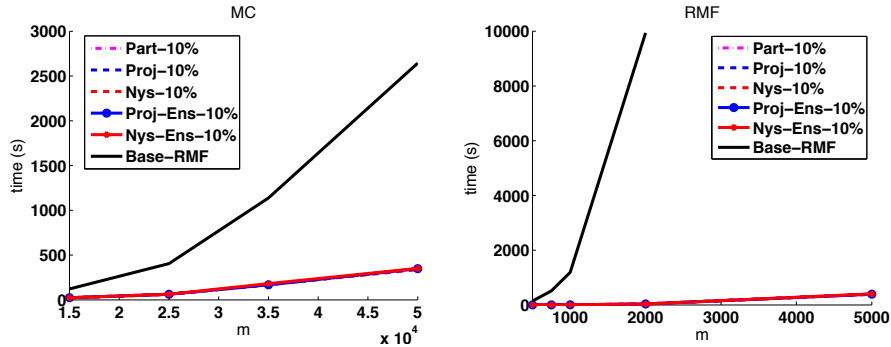

Figure 2: Speed-up of DFC relative to base algorithms.

## 3.2 Collaborative Filtering

Collaborative filtering for recommender systems is one prevalent real-world application of noisy matrix completion. A collaborative filtering dataset can be interpreted as the incomplete observation of a ratings matrix with columns corresponding to users and rows corresponding to items. The goal is to infer the unobserved entries of this ratings matrix. We evaluate DFC on two of the largest publicly available collaborative filtering datasets: MovieLens 10M[3] ($m = 4K, n = 6K, s > 10M$) and the Netflix Prize dataset[4] ($m = 18K, n = 480K, s > 100M$). To generate test sets drawn from the training distribution, for each dataset, we aggregated all available rating data into a single training set and withheld test entries uniformly at random, while ensuring that at least one training observation remained in each row and column. The algorithms were then run on the remaining training portions and evaluated on the test portions of each split. The results, averaged over three train-test splits, are summarized in Table 3.2. Notably, DFC-PROJ, DFC-PROJ-ENS, and DFC-NYS-ENS all outperform PARTITION, and DFC-PROJ-ENS performs comparably to APG while providing a nearly linear parallel time speed-up. The poorer performance of DFC-NYS can be in part explained by the asymmetry of these problems. Since these matrices have many more columns than rows, MF on column submatrices is inherently easier than MF on row submatrices, and for DFC-NYS, we observe that $\hat{\mathbf{C}}$ is an accurate estimate while $\hat{\mathbf{R}}$ is not.

Table 1: Performance of DFC relative to APG on collaborative filtering tasks.

| Method | MovieLens 10M | | Netflix | |
| --- | --- | --- | --- | --- |
| | RMSE | Time | RMSE | Time |
| APG | 0.8005 | 294.3s | 0.8433 | 2653.1s |
| PARTITION-25% | 0.8146 | 77.4s | 0.8451 | 689.1s |
| PARTITION-10% | 0.8461 | 36.0s | 0.8492 | 289.2s |
| DFC-NYS-25% | 0.8449 | 77.2s | 0.8832 | 890.9s |
| DFC-NYS-10% | 0.8769 | 53.4s | 0.9224 | 487.6s |
| DFC-NYS-ENS-25% | 0.8085 | 84.5s | 0.8486 | 964.3s |
| DFC-NYS-ENS-10% | 0.8327 | 63.9s | 0.8613 | 546.2s |
| DFC-PROJ-25% | 0.8061 | 77.4s | 0.8436 | 689.5s |
| DFC-PROJ-10% | 0.8272 | 36.1s | 0.8484 | 289.7s |
| DFC-PROJ-ENS-25% | 0.7944 | 77.4s | 0.8411 | 689.5s |
| DFC-PROJ-ENS-10% | 0.8119 | 36.1s | 0.8433 | 289.7s |

## 3.3 Background Modeling

Background modeling has important practical ramifications for detecting activity in surveillance video. This problem can be framed as an application of noisy RMF, where each video frame is a column of some matrix ($\mathbf{M}$), the background model is low-rank ($\mathbf{L}_0$), and moving objects and

background variations, e.g., changes in illumination, are outliers ($\mathbf{S}_0$). We evaluate DFC on two videos: 'Hall' (200 frames of size $176 \times 144$) contains significant foreground variation and was studied by [2], while 'Lobby' (1546 frames of size $168 \times 120$) includes many changes in illumination (a smaller video with 250 frames was studied by [2]). We focused on DFC-PROJ-ENS, due to its superior performance in previous experiments, and measured the RMSE between the background model recovered by DFC and that of APG. On both videos, DFC-PROJ-ENS recovered nearly the same background model as the full APG algorithm in a small fraction of the time. On 'Hall,' the DFC-PROJ-ENS-5% and DFC-PROJ-ENS-0.5% models exhibited RMSEs of $0.564$ and $1.55$, quite small given pixels with 256 intensity values. The associated runtime was reduced from $342.5s$ for APG to real-time ($5.2s$ for a 13s video) for DFC-PROJ-ENS-0.5%. Snapshots of the results are presented in Figure 3. On 'Lobby,' the RMSE of DFC-PROJ-ENS-4% was $0.64$, and the speed-up over APG was more than 20X, i.e., the runtime reduced from $16557s$ to $792s$.

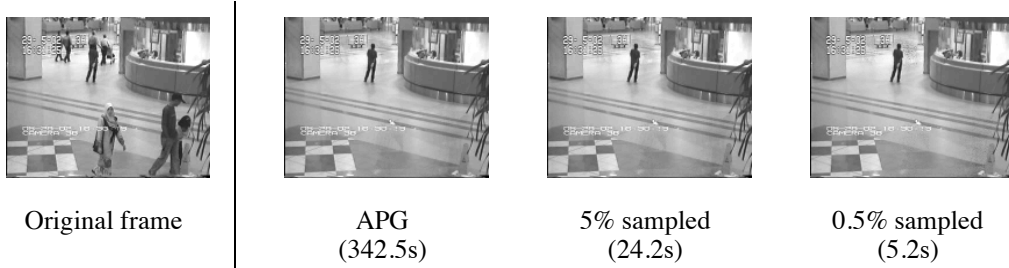

| Original frame | APG<br>(342.5s) | 5% sampled<br>(24.2s) | 0.5% sampled<br>(5.2s) |

Figure 3: Sample 'Hall' recovery by APG, DFC-PROJ-ENS-5%, and DFC-PROJ-ENS-.5%.

## 4 Theoretical Analysis

Having investigated the empirical advantages of DFC, we now show that DFC admits high-probability recovery guarantees comparable to those of its base algorithm.

### 4.1 Matrix Coherence

Since not all matrices can be recovered from missing entries or gross outliers, recent theoretical advances have studied sufficient conditions for accurate noisy MC [3, 12, 20] and RMF [1, 25]. Most prevalent among these are *matrix coherence* conditions, which limit the extent to which the singular vectors of a matrix are correlated with the standard basis. Letting $\mathbf{e}_i$ be the $i$th column of the standard basis, we define two standard notions of coherence [22]:

**Definition 1** ($\mu_0$-Coherence). *Let $\mathbf{V} \in \mathbb{R}^{n \times r}$ contain orthonormal columns with $r \leq n$. Then the $\mu_0$-coherence of $\mathbf{V}$ is:*

$$\mu_0(\mathbf{V}) \triangleq \frac{n}{r} \max_{1 \leq i \leq n} \|\mathbf{P}_V \mathbf{e}_i\|^2 = \frac{n}{r} \max_{1 \leq i \leq n} \|\mathbf{V}_{(i)}\|^2 .$$

**Definition 2** ($\mu_1$-Coherence). *Let $\mathbf{L} \in \mathbb{R}^{m \times n}$ have rank $r$. Then, the $\mu_1$-coherence of $\mathbf{L}$ is:*

$$\mu_1(\mathbf{L}) \triangleq \sqrt{\frac{mn}{r}} \max_{ij} |\mathbf{e}_i^\top \mathbf{U}_L \mathbf{V}_L^\top \mathbf{e}_j| .$$

For any $\mu > 0$, we will call a matrix $\mathbf{L}$ ($\mu, r$)-*coherent* if $\text{rank}(\mathbf{L}) = r$, $\max(\mu_0(\mathbf{U}_L), \mu_0(\mathbf{V}_L)) \leq \mu$, and $\mu_1(\mathbf{L}) \leq \sqrt{\mu}$. Our analysis will focus on base MC and RMF algorithms that express their recovery guarantees in terms of the ($\mu, r$)-coherence of the target low-rank matrix $\mathbf{L}_0$. For such algorithms, lower values of $\mu$ correspond to better recovery properties.

### 4.2 DFC Master Theorem

We now show that the same coherence conditions that allow for accurate MC and RMF also imply high-probability recovery for DFC. To make this precise, we let $\mathbf{M} = \mathbf{L}_0 + \mathbf{S}_0 + \mathbf{Z}_0 \in \mathbb{R}^{m \times n}$, where $\mathbf{L}_0$ is ($\mu, r$)-coherent and $\|\mathcal{P}_\Omega(\mathbf{Z}_0)\|_F \leq \Delta$. We further fix any $\epsilon, \delta \in (0, 1]$ and define $A(\mathbf{X})$ as the event that a matrix $\mathbf{X}$ is $(\frac{r\mu^2}{1-\epsilon/2}, r)$-coherent. Then, our Thm. 3 provides a generic recovery bound for DFC when used in combination with an arbitrary base algorithm. The proof requires a novel, coherence-based analysis of column projection and random column sampling. These results of independent interest are presented in Appendix A.

**Theorem 3.** *Choose $t = n/l$ and $l \geq cr\mu \log(n) \log(2/\delta)/\epsilon^2$, where $c$ is a fixed positive constant, and fix any $c_e \geq 0$. Under the notation of Algorithm 1, if a base MF algorithm yields $\mathbf{P}\left( \|\mathbf{C}_{0,i} - \hat{\mathbf{C}}_i\|_F > c_e\sqrt{ml}\Delta \mid A(\mathbf{C}_{0,i}) \right) \leq \delta_C$ for each $i$, where $\mathbf{C}_{0,i}$ is the corresponding partition of $\mathbf{L}_0$, then, with probability at least $(1-\delta)(1 - t\delta_C)$, DFC-PROJ guarantees*

$$\|\mathbf{L}_0 - \hat{\mathbf{L}}^{proj}\|_F \leq (2 + \epsilon)c_e\sqrt{mn}\Delta.$$

*Under Algorithm 2, if a base MF algorithm yields $\mathbf{P}\left( \|\mathbf{C}_0 - \hat{\mathbf{C}}\|_F > c_e\sqrt{ml}\Delta \mid A(\mathbf{C}) \right) \leq \delta_C$ and $\mathbf{P}\left( \|\mathbf{R}_0 - \hat{\mathbf{R}}\|_F > c_e\sqrt{dn}\Delta \mid A(\mathbf{R}) \right) \leq \delta_R$ for $d \geq cl\mu_0(\hat{\mathbf{C}}) \log(m) \log(1/\delta)/\epsilon^2$, then, with probability at least $(1-\delta)^2(1 - \delta_C - \delta_R)$, DFC-NYS guarantees*

$$\|\mathbf{L}_0 - \hat{\mathbf{L}}^{nys}\|_F \leq (2 + 3\epsilon)c_e\sqrt{ml + dn}\Delta.$$

To understand the conclusions of Thm. 3, consider a typical base algorithm which, when applied to $\mathcal{P}_\Omega(\mathbf{M})$, recovers an estimate $\hat{\mathbf{L}}$ satisfying $\|\mathbf{L}_0 - \hat{\mathbf{L}}\|_F \leq c_e\sqrt{mn}\Delta$ with high probability. Thm. 3 asserts that, with appropriately reduced probability, DFC-PROJ exhibits the same recovery error scaled by an adjustable factor of $2 + \epsilon$, while DFC-NYS exhibits a somewhat smaller error scaled by $2 + 3\epsilon$.[5] The key take-away then is that DFC introduces a controlled increase in error and a controlled decrement in the probability of success, allowing the user to interpolate between maximum speed and maximum accuracy. Thus, DFC can quickly provide near-optimal recovery in the noisy setting and exact recovery in the noiseless setting ($\Delta = 0$), even when entries are missing or grossly corrupted. The next two sections demonstrate how Thm. 3 can be applied to derive specific DFC recovery guarantees for noisy MC and noisy RMF. In these sections, we let $\bar{n} \triangleq \max(m, n)$.

### 4.3 Consequences for Noisy MC

Our first corollary of Thm. 3 shows that DFC retains the high-probability recovery guarantees of a standard MC solver while operating on matrices of much smaller dimension. Suppose that a base MC algorithm solves the following convex optimization problem, studied in [3]:

$$\text{minimize}_{\mathbf{L}} \quad \|\mathbf{L}\|_* \quad \text{subject to} \quad \|\mathcal{P}_\Omega(\mathbf{M} - \mathbf{L})\|_F \leq \Delta.$$

Then, Cor. 4 follows from a novel guarantee for noisy convex MC, proved in the appendix.

**Corollary 4.** *Suppose that $\mathbf{L}_0$ is $(\mu, r)$-coherent and that $s$ entries of $\mathbf{M}$ are observed, with locations $\Omega$ distributed uniformly. Define the oversampling parameter*

$$\beta_s \triangleq \frac{s(1 - \epsilon/2)}{32\mu^2 r^2 (m+n) \log^2(m+n)},$$

*and fix any target rate parameter $1 < \beta \leq \beta_s$. Then, if $\|\mathcal{P}_\Omega(\mathbf{M}) - \mathcal{P}_\Omega(\mathbf{L}_0)\|_F \leq \Delta$ a.s., it suffices to choose $t = n/l$ and*

$$l \geq \max\left( \frac{n\beta}{\beta_s} + \sqrt{\frac{n(\beta-1)}{\beta_s}}, cr\mu\frac{\log(n)\log(2/\delta)}{\epsilon^2} \right), \quad d \geq \max\left( \frac{m\beta}{\beta_s} + \sqrt{\frac{m(\beta-1)}{\beta_s}}, cl\mu_0(\hat{\mathbf{C}})\frac{\log(m)\log(1/\delta)}{\epsilon^2} \right)$$

*to achieve*

> **DFC-PROJ:** $\|\mathbf{L}_0 - \hat{\mathbf{L}}^{proj}\|_F \leq (2 + \epsilon)c_e'\sqrt{mn}\Delta$

> **DFC-NYS:** $\|\mathbf{L}_0 - \hat{\mathbf{L}}^{nys}\|_F \leq (2 + 3\epsilon)c_e'\sqrt{ml + dn}\Delta$

*with probability at least*

> **DFC-PROJ:** $(1-\delta)(1 - 5t\log(\bar{n})\bar{n}^{2-2\beta}) \geq (1-\delta)(1 - \bar{n}^{3-2\beta})$

> **DFC-NYS:** $(1-\delta)^2(1 - 10\log(\bar{n})\bar{n}^{2-2\beta})$,

*respectively, with $c$ as in Thm. 3 and $c_e'$ a positive constant.*

Notably, Cor. 4 allows for the fraction of columns and rows sampled to decrease as the oversampling parameter $\beta_s$ increases with $m$ and $n$. In the best case, $\beta_s = \Theta(mn/[(m+n)\log^2(m+n)])$, and Cor. 4 requires only $O(\frac{n}{m}\log^2(m+n))$ sampled columns and $O(\frac{m}{n}\log^2(m+n))$ sampled rows. In the worst case, $\beta_s = \Theta(1)$, and Cor. 4 requires the number of sampled columns and rows to grow linearly with the matrix dimensions. As a more realistic intermediate scenario, consider the setting in which $\beta_s = \Theta(\sqrt{m+n})$ and thus a vanishing fraction of entries are revealed. In this setting, only $O(\sqrt{m+n})$ columns and rows are required by Cor. 4.

### 4.4 Consequences for Noisy RMF

Our next corollary shows that DFC retains the high-probability recovery guarantees of a standard RMF solver while operating on matrices of much smaller dimension. Suppose that a base RMF algorithm solves the following convex optimization problem, studied in [25]:

$$\text{minimize}_{\mathbf{L},\mathbf{S}} \quad \|\mathbf{L}\|_* + \lambda\|\mathbf{S}\|_1 \quad \text{subject to} \quad \|\mathbf{M}-\mathbf{L}-\mathbf{S}\|_F \leq \Delta,$$

with $\lambda = 1/\sqrt{\bar{n}}$. Then, Cor. 5 follows from Thm. 3 and the noisy RMF guarantee of [25, Thm. 2].

**Corollary 5.** *Suppose that $\mathbf{L}_0$ is $(\mu,r)$-coherent and that the uniformly distributed support set of $\mathbf{S}_0$ has cardinality $s$. For a fixed positive constant $\rho_s$, define the undersampling parameter*

$$\beta_s \triangleq \left(1 - \frac{s}{mn}\right)/\rho_s,$$

*and fix any target rate parameter $\beta > 2$ with rescaling $\beta' \triangleq \beta\log(\bar{n})/\log(m)$ satisfying $4\beta_s - 3/\rho_s \leq \beta' \leq \beta_s$. Then, if $\|\mathbf{M}-\mathbf{L}_0-\mathbf{S}_0\|_F \leq \Delta$ a.s., it suffices to choose $t = n/l$ and*

$$l \geq \max\left(\frac{r^2\mu^2\log^2(\bar{n})}{(1-\epsilon/2)\rho_r}, \frac{4\log(\bar{n})\beta(1-\rho_s\beta_s)}{m(\rho_s\beta_s - \rho_s\beta')^2}, cr\mu\log(n)\log(2/\delta)/\epsilon^2\right)$$

$$d \geq \max\left(\frac{r^2\mu^2\log^2(\bar{n})}{(1-\epsilon/2)\rho_r}, \frac{4\log(\bar{n})\beta(1-\rho_s\beta_s)}{n(\rho_s\beta_s - \rho_s\beta')^2}, cl\mu_0(\hat{\mathbf{C}})\log(m)\log(1/\delta)/\epsilon^2\right)$$

*to have*

**DFC-PROJ:** $\|\mathbf{L}_0 - \hat{\mathbf{L}}^{proj}\|_F \leq (2+\epsilon)c_e''\sqrt{mn}\Delta$

**DFC-NYS:** $\|\mathbf{L}_0 - \hat{\mathbf{L}}^{nys}\|_F \leq (2+3\epsilon)c_e''\sqrt{ml+dn}\Delta$

*with probability at least*

**DFC-PROJ:** $(1-\delta)(1-tc_p\bar{n}^{-\beta}) \geq (1-\delta)(1-c_p\bar{n}^{1-\beta})$

**DFC-NYS:** $(1-\delta)^2(1-2c_p\bar{n}^{-\beta}),$

*respectively, with $c$ as in Thm. 3 and $\rho_r, c_e''$, and $c_p$ positive constants.*

Note that Cor. 5 places only very mild restrictions on the number of columns and rows to be sampled. Indeed, $l$ and $d$ need only grow poly-logarithmically in the matrix dimensions to achieve high-probability noisy recovery.

## 5 Conclusions

To improve the scalability of existing matrix factorization algorithms while leveraging the ubiquity of parallel computing architectures, we introduced, evaluated, and analyzed DFC, a divide-and-conquer framework for noisy matrix factorization with missing entries or outliers. We note that the contemporaneous work of [19] addresses the computational burden of noiseless RMF by reformulating a standard convex optimization problem to internally incorporate random projections. The differences between DFC and the approach of [19] highlight some of the main advantages of this work: i) DFC can be used in combination with any underlying MF algorithm, ii) DFC is trivially parallelized, and iii) DFC provably maintains the recovery guarantees of its base algorithm, even in the presence of noise.

## Footnotes

[1] For ease of discussion, we assume that $\mathrm{mod}(n, t) = 0$, and hence, $l = n/t$. Note that for arbitrary $n$ and $t$, $\mathcal{P}_\Omega(\mathbf{M})$ can always be partitioned into $t$ submatrices, each with either $\lfloor n/t \rfloor$ or $\lceil n/t \rceil$ columns.

[2]In the left-hand plot of Figure 1, the lines for Proj-10% and Proj-Ens-10% overlap.

[3] http://www.grouplens.org/

[4] http://www.netflixprize.com/

[5]Note that the DFC-NYS guarantee requires the number of rows sampled to grow in proportion to $\mu_0(\hat{\mathbf{C}})$, a quantity always bounded by $\mu$ in our simulations.

# References

[1] A. Agarwal, S. Negahban, and M. J. Wainwright. Noisy matrix decomposition via convex relaxation: Optimal rates in high dimensions. In *International Conference on Machine Learning*, 2011.

[2] E. J. Candès, X. Li, Y. Ma, and J. Wright. Robust principal component analysis? *Journal of the ACM*, 58 (3):1–37, 2011.

[3] E.J. Candès and Y. Plan. Matrix completion with noise. *Proceedings of the IEEE*, 98(6):925 –936, 2010.

[4] V. Chandrasekaran, S. Sanghavi, P. A. Parrilo, and A. S. Willsky. Sparse and low-rank matrix decompositions. In *Allerton Conference on Communication, Control, and Computing*, 2009.

[5] Y. Chen, H. Xu, C. Caramanis, and S. Sanghavi. Robust matrix completion and corrupted columns. In *International Conference on Machine Learning*, 2011.

[6] P. Drineas, M. W. Mahoney, and S. Muthukrishnan. Relative-error CUR matrix decompositions. *SIAM Journal on Matrix Analysis and Applications*, 30:844–881, 2008.

[7] A. Frieze, R. Kannan, and S. Vempala. Fast Monte-Carlo algorithms for finding low-rank approximations. In *Foundations of Computer Science*, 1998.

[8] S. A. Goreinov, E. E. Tyrtyshnikov, and N. L. Zamarashkin. A theory of pseudoskeleton approximations. *Linear Algebra and its Applications*, 261(1-3):1 – 21, 1997.

[9] D. Gross and V. Nesme. Note on sampling without replacing from a finite collection of matrices. *CoRR*, abs/1001.2738, 2010.

[10] W. Hoeffding. Probability inequalities for sums of bounded random variables. *Journal of the American Statistical Association*, 58(301):13–30, 1963.

[11] D. Hsu, S. M. Kakade, and T. Zhang. Dimension-free tail inequalities for sums of random matrices. `arXiv:1104.1672v3[math.PR]`, 2011.

[12] R. H. Keshavan, A. Montanari, and S. Oh. Matrix completion from noisy entries. *Journal of Machine Learning Research*, 99:2057–2078, 2010.

[13] S. Kumar, M. Mohri, and A. Talwalkar. On sampling-based approximate spectral decomposition. In *International Conference on Machine Learning*, 2009.

[14] S. Kumar, M. Mohri, and A. Talwalkar. Ensemble Nyström method. In *NIPS*, 2009.

[15] Z. Lin, A. Ganesh, J. Wright, L. Wu, M. Chen, and Y. Ma. Fast convex optimization algorithms for exact recovery of a corrupted low-rank matrix. UIUC Technical Report UILU-ENG-09-2214, 2009.

[16] S. Ma, D. Goldfarb, and L. Chen. Fixed point and bregman iterative methods for matrix rank minimization. *Mathematical Programming*, 128(1-2):321–353, 2011.

[17] K. Min, Z. Zhang, J. Wright, and Y. Ma. Decomposing background topics from keywords by principal component pursuit. In *Conference on Information and Knowledge Management*, 2010.

[18] M. Mohri and A. Talwalkar. Can matrix coherence be efficiently and accurately estimated? In *Conference on Artificial Intelligence and Statistics*, 2011.

[19] Y. Mu, J. Dong, X. Yuan, and S. Yan. Accelerated low-rank visual recovery by random projection. In *Conference on Computer Vision and Pattern Recognition*, 2011.

[20] S. Negahban and M. J. Wainwright. Restricted strong convexity and weighted matrix completion: Optimal bounds with noise. `arXiv:1009.2118v2[cs.IT]`, 2010.

[21] Y. Peng, A. Ganesh, J. Wright, W. Xu, and Y. Ma. Rasl: Robust alignment by sparse and low-rank decomposition for linearly correlated images. In *Conference on Computer Vision and Pattern Recognition*, 2010.

[22] B. Recht. A simpler approach to matrix completion. `arXiv:0910.0651v2[cs.IT]`, 2009.

[23] K. Toh and S. Yun. An accelerated proximal gradient algorithm for nuclear norm regularized least squares problems. *Pacific Journal of Optimization*, 6(3):615–640, 2010.

[24] C.K. Williams and M. Seeger. Using the Nyström method to speed up kernel machines. In *NIPS*, 2000.

[25] Z. Zhou, X. Li, J. Wright, E. J. Candès, and Y. Ma. Stable principal component pursuit. `arXiv: 1001.2363v1[cs.IT]`, 2010.

